# Fast Graph Laplacian Regularized Kernel Learning via Semidefinite–Quadratic–Linear Programming

**Xiao-Ming Wu**
Dept. of IE
The Chinese University of Hong Kong
wxm007@ie.cuhk.edu.hk

**Anthony Man-Cho So**
Dept. of SE&EM
The Chinese University of Hong Kong
manchoso@se.cuhk.edu.hk

**Zhenguo Li**
Dept. of IE
The Chinese University of Hong Kong
zgli@ie.cuhk.edu.hk

**Shuo-Yen Robert Li**
Dept. of IE
The Chinese University of Hong Kong
bobli@ie.cuhk.edu.hk

## Abstract

Kernel learning is a powerful framework for nonlinear data modeling. Using the kernel trick, a number of problems have been formulated as semidefinite programs (SDPs). These include Maximum Variance Unfolding (MVU) (Weinberger et al., 2004) in nonlinear dimensionality reduction, and Pairwise Constraint Propagation (PCP) (Li et al., 2008) in constrained clustering. Although in theory SDPs can be efficiently solved, the high computational complexity incurred in numerically processing the huge linear matrix inequality constraints has rendered the SDP approach unscalable. In this paper, we show that a large class of kernel learning problems can be reformulated as semidefinite-quadratic-linear programs (SQLPs), which only contain a simple positive semidefinite constraint, a second-order cone constraint and a number of *linear* constraints. These constraints are much easier to process numerically, and the gain in speedup over previous approaches is at least of the order $m^{2.5}$, where $m$ is the matrix dimension. Experimental results are also presented to show the superb computational efficiency of our approach.

## 1 Introduction

Kernel methods provide a principled framework for nonlinear data modeling, where the inference in the input space can be transferred intactly to any feature space by simply treating the associated kernel as inner products, or more generally, as nonlinear mappings on the data (Schölkopf & Smola, 2002). Some well-known kernel methods include support vector machines (SVMs) (Vapnik, 2000), kernel principal component analysis (kernel PCA) (Schölkopf et al., 1998), and kernel $k$-means (Shawe-Taylor & Cristianini, 2004). Naturally, an important issue in kernel methods is kernel design. Indeed, the performance of a kernel method depends crucially on the kernel used, where different choices of kernels often lead to quite different results. Therefore, substantial efforts have been made to design appropriate kernels for the problems at hand. For instance, in (Chapelle & Vapnik, 2000), parametric kernel functions are proposed, where the focus is on model selection (Chapelle & Vapnik, 2000). The modeling capability of parametric kernels, however, is limited. A more natural idea is to learn specialized nonparametric kernels for specific problems. For instance, in cases where only inner products of the input data are involved, kernel learning is equivalent to the learning of a kernel matrix. This is the main focus of recent kernel methods.

Currently, many different kernel learning frameworks have been proposed. These include spectral kernel learning (Li & Liu, 2009), multiple kernel learning (Lanckriet et al., 2004), and the Breg-

man divergence-based kernel learning (Kulis et al., 2009). Typically, a kernel learning framework consists of two main components: the problem formulation in terms of the kernel matrix, and an optimization procedure for finding the kernel matrix that has certain desirable properties. As seen in, e.g., the *Maximum Variance Unfolding* (MVU) method (Weinberger et al., 2004) for nonlinear dimensionality reduction (see (So, 2007) for related discussion) and *Pairwise Constraint Propagation* (PCP) (Li et al., 2008) for constrained clustering, a nice feature of such a framework is that the problem formulation often becomes straightforward. Thus, the major challenge in optimization-based kernel learning lies in the second component, where the key is to find an efficient procedure to obtain a positive semidefinite kernel matrix that satisfies certain properties.

Using the kernel trick, most kernel learning problems (Graepel, 2002; Weinberger et al., 2004; Globerson & Roweis, 2007; Song et al., 2008; Li et al., 2008) can naturally be formulated as semidefinite programs (SDPs). Although in theory SDPs can be efficiently solved, the high computational complexity has rendered the SDP approach unscalable. An effective and widely used heuristic for speedup is to perform low-rank kernel approximation and matrix factorization (Weinberger et al., 2005; Weinberger et al., 2007; Li et al., 2009). In this paper, we investigate the possibility of further speedup by studying a class of convex quadratic semidefinite programs (QSDPs). These QSDPs arise in many contexts, such as graph Laplacian regularized low-rank kernel learning in nonlinear dimensionality reduction (Sha & Saul, 2005; Weinberger et al., 2007; Globerson & Roweis, 2007; Song et al., 2008; Singer, 2008) and constrained clustering (Li et al., 2009). In the aforementioned papers, a QSDP is reformulated as an SDP with $O(m^2)$ variables and a linear matrix inequality of size $O(m^2) \times O(m^2)$. Such a reformulation is highly inefficient and unscalable, as it has an order of $m^9$ time complexity (Ben-Tal & Nemirovski, 2001, Lecture 6). In this paper, we propose a novel reformulation that exploits the structure of the QSDP and leads to a semidefinite-quadratic-linear program (SQLP) that can be solved by the standard software SDPT3 (Tütüncü et al., 2003). Such a reformulation has the advantage that it only has one positive semidefinite constraint on a matrix of size $m \times m$, one second-order cone constraint of size $O(m^2)$ and a number of *linear* constraints on $O(m^2)$ variables. As a result, our reformulation is much easier to process numerically. Moreover, a simple complexity analysis shows that the gain in speedup over previous approaches is at least of the order $m^{2.5}$. Experimental results show that our formulation is indeed far more efficient than previous ones.

The rest of the paper is organized as follows. We review related kernel learning problems in Section 2 and present our formulation in Section 3. Experiment results are reported in Section 4. Section 5 concludes the paper.

## 2   The Problems

In this section, we briefly review some kernel learning problems that arise in dimensionality reduction and constrained clustering. They include MVU (Weinberger et al., 2004), Colored MVU (Song et al., 2008), (Singer, 2008), Pairwise Semidefinite Embedding (PSDE) (Globerson & Roweis, 2007), and PCP (Li et al., 2008). MVU maximizes the variance of the embedding while preserving local distances of the input data. Colored MVU generalizes MVU with side information such as class labels. PSDE derives an embedding that strictly respects known similarities, in the sense that objects known to be similar are always closer in the embedding than those known to be dissimilar. PCP is designed for constrained clustering, which embeds the data on the unit hypersphere such that two objects that are known to be from the same cluster are embedded to the same point, while two objects that are known to be from different clusters are embedded orthogonally. In particular, PCP seeks the smoothest mapping for such an embedding, thereby propagating pairwise constraints.

Initially, each of the above problems is formulated as an SDP, whose kernel matrix $K$ is of size $n \times n$, where $n$ denotes the number of objects. Since such an SDP is computationally expensive, one can try to reduce the problem size by using graph Laplacian regularization. In other words, one takes $K = QYQ^T$, where $Q \in \mathbb{R}^{n \times m}$ consists of the smoothest $m$ eigenvectors of the graph Laplacian ($m \ll n$), and $Y$ is of size $m \times m$ (Sha & Saul, 2005; Weinberger et al., 2007; Song et al., 2008; Globerson & Roweis, 2007; Singer, 2008; Li et al., 2009). The learning of $K$ is then reduced to the learning of $Y$, leading to a quadratic semidefinite program (QSDP) that is similar to a standard quadratic program (QP), except that the feasible set of a QSDP resides in the positive semidefinite cone as well. The intuition behind this low-rank kernel approximation is that a kernel matrix of the

form $K = QYQ^T$ actually, to some degree, preserves the proximity of objects in the feature space. Detailed justification can be found in the related work mentioned above.

Next, we use MVU and PCP as representatives to demonstrate how the SDP formulations emerge from nonlinear dimensionality reduction and constrained clustering.

## 2.1 MVU

The SDP of MVU (Weinberger et al., 2004) is as follows:

$$\max_{K} \ \text{tr}(K) = I \bullet K \tag{1}$$

$$\text{s.t.} \ \sum_{i,j=1}^{n} k_{ij} = 0, \tag{2}$$

$$k_{ii} + k_{jj} - 2k_{ij} = d_{ij}^2, \ \forall (i,j) \in \mathcal{N}, \tag{3}$$

$$K \succeq 0, \tag{4}$$

where $K = (k_{ij})$ denotes the kernel matrix to be learned, $I$ denotes the identity matrix, $\text{tr}(\cdot)$ denotes the trace of a square matrix, $\bullet$ denotes the element-wise dot product between matrices, $d_{ij}$ denotes the Euclidean distance between the $i$-th and $j$-th objects, and $\mathcal{N}$ denotes the set of neighbor pairs, whose distances are to be preserved in the embedding.

The constraint in (2) centers the embedding at the origin, thus removing the translation freedom. The constraints in (3) preserve local distances. The constraint $K \succeq 0$ in (4) specifies that $K$ must be symmetric and positive semidefinite, which is necessary since $K$ is taken as the inner product matrix of the embedding. Note that given the constraint in (2), the variance of the embedding is characterized by $\mathcal{V}(K) = \frac{1}{2n} \sum_{i,j} (k_{ii} + k_{jj} - 2k_{ij}) = \text{tr}(K)$ (Weinberger et al., 2004) (see related discussion in (So, 2007), Chapter 4). Thus, the SDP in (1-4) maximizes the variance of the embedding while keeping local distances unchanged. After $K$ is obtained, kernel PCA is applied to $K$ to compute the low-dimensional embedding.

## 2.2 PCP

The SDP of PCP (Li et al., 2008) is:

$$\min_{K} \ \bar{L} \bullet K \tag{5}$$

$$\text{s.t.} \ k_{ii} = 1, \ i = 1, 2, \ldots, n, \tag{6}$$

$$k_{ij} = 1, \ \forall (i,j) \in \mathcal{M}, \tag{7}$$

$$k_{ij} = 0, \ \forall (i,j) \in \mathcal{C}, \tag{8}$$

$$K \succeq 0, \tag{9}$$

where $\bar{L}$ denotes the normalized graph Laplacian, $\mathcal{M}$ denotes the set of object pairs that are known to be from the same cluster, and $\mathcal{C}$ denotes those that are known to be from different clusters.

The constraints in (6) map the objects to the unit hypersphere. The constraints in (7) map two objects that are known to be from the same cluster to the same vector. The constraints in (8) map two objects that are known to be from different clusters to vectors that are orthogonal. Let $\mathcal{X} = \{\mathbf{x}_i\}_{i=1}^{n}$ be the data set, $\mathcal{F}$ be the feature space, and $\phi : \mathcal{X} \to \mathcal{F}$ be the associated feature map of $K$. Then, the degree of smoothness of $\phi$ on the data graph can be captured by (Zhou et al., 2004):

$$\mathcal{S}(\phi) = \frac{1}{2} \sum_{i,j=1}^{n} w_{ij} \left\| \frac{\phi(\mathbf{x}_i)}{\sqrt{d_{ii}}} - \frac{\phi(\mathbf{x}_j)}{\sqrt{d_{jj}}} \right\|_{\mathcal{F}}^{2} = \bar{L} \bullet K, \tag{10}$$

where $w_{ij}$ is the similarity of $\mathbf{x}_i$ and $\mathbf{x}_j$, $d_{ii} = \sum_{j=1}^{n} w_{ij}$, and $\| \cdot \|_{\mathcal{F}}$ is the distance metric in $\mathcal{F}$. The smaller the value $\mathcal{S}(\phi)$, the smoother is the feature map $\phi$. Thus, the SDP in (5-9) seeks the smoothest feature map that embeds the data on the unit hypersphere and at the same time respects the pairwise constraints. After $K$ is solved, kernel $k$-means is then applied to $K$ to obtain the clusters.

## 2.3 Low-Rank Approximation: from SDP to QSDP

The SDPs in MVU and PCP are difficult to solve efficiently because their computational complexity scales at least cubically in the size of the matrix variable and the number of constraints (Borchers, 1999). A useful heuristic is to use low-rank kernel approximation, which is motivated by the observation that the degree of freedom in the data is often much smaller than the number of parameters in a fully nonparametric kernel matrix $K$. For instance, it may be equal to or slightly larger than the intrinsic dimension of the data manifold (for dimensionality reduction) or the number of clusters (for clustering). Another more specific observation is that it is often desirable to have nearby objects mapped to nearby points, as is done in MVU or PCP.

Based on these observations, instead of learning a fully nonparametric $K$, one learns a $K$ of the form $K = QYQ^T$, where $Q$ and $Y$ are of sizes $n \times m$ and $m \times m$, respectively, where $m \ll n$. The matrix $Q$ should be smooth in the sense that nearby objects in the input space are mapped to nearby points (the $i$-th row of $Q$ is taken as a new representation of $\mathbf{x}_i$). $Q$ is computed prior to the learning of $K$. In this way, the learning of a kernel matrix $K$ is reduced to the learning of a much smaller $Y$, subject to the constraint that $Y \succeq 0$. This idea is used in (Weinberger et al., 2007) and (Li et al., 2009) to speed up MVU and PCP, respectively, and is also adopted in Colored MVU (Song et al., 2008) and PSDE (Globerson & Roweis, 2007) for efficient computation.

The choice of $Q$ can be different for MVU and PCP. In (Weinberger et al., 2007), $Q = (\mathbf{v}_2, \ldots, \mathbf{v}_{m+1})$, where $\{\mathbf{v}_i\}$ are the eigenvectors of the graph Laplacian. In (Li et al., 2009), $Q = (\mathbf{u}_1, \ldots, \mathbf{u}_m)$, where $\{\mathbf{u}_i\}$ are the eigenvectors of the normalized graph Laplacian. Since such $Q$'s are obtained from graph Laplacians, we call the learning of $K$ of the form $K = QYQ^T$ the *Graph Laplacian Regularized Kernel Learning* problem, and denote the methods in (Weinberger et al., 2007) and (Li et al., 2009) by RegMVU and RegPCP, respectively.

With $K = QYQ^T$, RegMVU and RegPCP become:

$$\text{RegMVU}: \ \max_{Y \succeq 0} \ \text{tr}(Y) - \nu \sum_{(i,j) \in \mathcal{N}} ((QYQ^T)_{ii} - 2(QYQ^T)_{ij} + (QYQ^T)_{jj} - d_{ij}^2)^2, \quad (11)$$

$$\text{RegPCP}: \ \min_{Y \succeq 0} \ \sum_{(i,j,t_{ij}) \in \mathcal{S}} ((QYQ^T)_{ij} - t_{ij})^2, \quad (12)$$

where $\nu > 0$ is a regularization parameter and $\mathcal{S} = \{(i,j,t_{ij}) \mid (i,j) \in \mathcal{M} \cup \mathcal{C}, \text{or } i = j, t_{ij} = 1 \text{ if } (i,j) \in \mathcal{M} \text{ or } i = j, t_{ij} = 0 \text{ if } (i,j) \in \mathcal{C}\}$. Both RegMVU and RegPCP can be succinctly rewritten in the unified form:

$$\min_{\mathbf{y}} \ \mathbf{y}^T A \mathbf{y} + \mathbf{b}^T \mathbf{y} \quad (13)$$

$$\text{s.t. } Y \succeq 0, \quad (14)$$

where $\mathbf{y} = \text{vec}(Y) \in \mathbb{R}^{m^2}$ denotes the vector obtained by concatenating all the columns of $Y$, and $A \succeq 0$ (Weinberger et al., 2007; Li et al., 2009). Note that this problem is convex since both the objective function and the feasible set are convex.

Problem (13-14) is an instance of the so-called *convex quadratic semidefinite program* (QSDP), where the objective is a quadratic function in the matrix variable $Y$. Note that similar QSDPs arise in Colored MVU, PSDE, Conformal Eigenmaps (Sha & Saul, 2005), Locally Rigid Embedding (Singer, 2008), and Kernel Matrix Completion (Graepel, 2002). Before we present our approach for tackling the QSDP (13-14), let us briefly review existing approaches in the literature.

## 2.4 Previous Approach: from QSDP to SDP

Currently, a typical approach for tackling a QSDP is to use the Schur complement (Boyd & Vandenberghe, 2004) to rewrite it as an SDP (Sha & Saul, 2005; Weinberger et al., 2007; Li et al., 2009; Song et al., 2008; Globerson & Roweis, 2007; Singer, 2008; Graepel, 2002), and then solve it using an SDP solver such as CSDP[1] (Borchers, 1999) or SDPT3[2] (Toh et al., 2006). In this paper, we call

this approach the *Schur Complement Based SDP* (SCSDP) formulation. For the QSDP in (13-14), the equivalent SDP takes the form:

$$\min_{\mathbf{y},\nu} \; \nu + \mathbf{b}^T \mathbf{y} \qquad (15)$$

$$\text{s.t. } Y \succeq 0 \;\text{ and } \begin{bmatrix} I_{m^2} & A^{\frac{1}{2}}\mathbf{y} \\ (A^{\frac{1}{2}}\mathbf{y})^T & \nu \end{bmatrix} \succeq 0, \qquad (16)$$

where $A^{\frac{1}{2}}$ is the matrix square root of $A$, $I_{m^2}$ is the identity matrix of size $m^2 \times m^2$, and $\nu$ is a slack variable serving as an upper bound of $\mathbf{y}^T A \mathbf{y}$. The second semidefinite cone constraint is equivalent to $(A^{\frac{1}{2}}\mathbf{y})^T (A^{\frac{1}{2}}\mathbf{y}) \leq \nu$ by the Schur complement.

Although the SDP in (15-16) has only $m(m+1)/2+1$ variables, it has two semidefinite cone constraints, of sizes $m \times m$ and $(m^2+1) \times (m^2+1)$, respectively. Such an SDP not only scales poorly, but is also difficult to process numerically. Indeed, by considering Problem (15-16) as an SDP in the standard dual form, the number of iterations required by standard interior-point algorithms is of the order $m$, and the total number of arithmetic operations required is of the order $m^9$ (Ben-Tal & Nemirovski, 2001, Lecture 6). In practice, it takes only a few seconds to solve the aforementioned SDP when $m = 10$, but can take more than 1 day when $m = 40$ (see Section 4 for details). Thus, it is not surprising that $m$ is set to a very small value in the related methods—for example, $m = 10$ in (Weinberger et al., 2007) and $m = 15$ in (Li et al., 2009). However, as noted by the authors in (Weinberger et al., 2007), a larger $m$ does lead to better performance. In (Li et al., 2009), the authors suggest that $m$ should be larger than the number of clusters.

Is this formulation from QSDP to SDP the best we can have? The answer is no. In the next section, we present a novel formulation that leads to a semidefinite-quadratic-linear program (SQLP), which is much more efficient and scalable than the one above. For instance, it takes about 15 seconds when $m = 30$, 2 minutes when $m = 40$, and 1 hour when $m = 100$, as reported in Section 4.

## 3 Our Formulation: from QSDP to SQLP

In this section, we formulate the QSDP in (13-14) as an SQLP. Though our focus here is on the QSDP in (13-14), we should point out that our method applies to any convex QSDP.

Recall that the size of $A$ is $m^2 \times m^2$. Let $r$ be the rank of $A$. With Cholesky factorization, we can obtain an $r \times m^2$ matrix $B$ such that $A = B^T B$, as $A$ is symmetric positive semidefinite and of rank $r$ (Golub & Loan, 1996). Now, let $\mathbf{z} = B\mathbf{y}$. Then, the QSDP in (13-14) is equivalent to:

$$\min_{\mathbf{y},\mathbf{z},\mu} \; \mu + \mathbf{b}^T \mathbf{y} \qquad (17)$$

$$\text{s.t. } \mathbf{z} = B\mathbf{y}, \qquad (18)$$

$$\mathbf{z}^T \mathbf{z} \leq \mu, \qquad (19)$$

$$Y \succeq 0. \qquad (20)$$

Next, we show that the constraint in (19) is equivalent to a second-order cone constraint. Let $\mathcal{K}_n$ be the second-order cone of dimension $n$, i.e.,

$$\mathcal{K}_n = \{(x_0; \mathbf{x}) \in \mathbb{R}^n : x_0 \geq \|\mathbf{x}\|\},$$

where $\|\cdot\|$ denotes the standard Euclidean norm. Let $\mathbf{u} = (\frac{1+\mu}{2}, \frac{1-\mu}{2}, \mathbf{z}^T)^T$. Then, the following holds.

**Theorem 3.1.** $\mathbf{z}^T \mathbf{z} \leq \mu$ *if and only if* $\mathbf{u} \in \mathcal{K}_{r+2}$.

*Proof.* Note that $\mathbf{u} \in \mathbb{R}^{r+2}$, since $\mathbf{z} \in \mathbb{R}^r$. Also, note that $\mu = (\frac{1+\mu}{2})^2 - (\frac{1-\mu}{2})^2$. If $\mathbf{z}^T \mathbf{z} \leq \mu$, then $(\frac{1+\mu}{2})^2 - (\frac{1-\mu}{2})^2 = \mu \geq \mathbf{z}^T \mathbf{z}$, which means that $\frac{1+\mu}{2} \geq \|(\frac{1-\mu}{2}, \mathbf{z}^T)^T\|$. In particular, we have $\mathbf{u} \in \mathcal{K}_{r+2}$. Conversely, if $\mathbf{u} \in \mathcal{K}_{r+2}$, then $(\frac{1+\mu}{2})^2 \geq (\frac{1-\mu}{2})^2 + \mathbf{z}^T \mathbf{z}$, thus implying $\mathbf{z}^T \mathbf{z} \leq \mu$. $\square$

Let $\mathbf{e}_i$ (where $i = 1, 2, \ldots, r+2$) be the $i$-th basis vector, and let $C = (\mathbf{0}_{r \times 2}, I_{r \times r})$. Then, we have $(\mathbf{e}_1 - \mathbf{e}_2)^T \mathbf{u} = \mu$, $(\mathbf{e}_1 + \mathbf{e}_2)^T \mathbf{u} = 1$, and $\mathbf{z} = C\mathbf{u}$. Hence, by Theorem 3.1, the problem in (17-20)

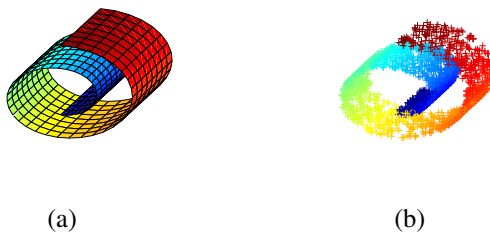

<center>(a)                        (b)</center>

Figure 1: Swiss Roll. (a) The true manifold. (b) A set of 2000 points sampled from the manifold.

is equivalent to:

$$\min_{\mathbf{y},\mathbf{u}} \ (\mathbf{e}_1 - \mathbf{e}_2)^T \mathbf{u} + \mathbf{b}^T \mathbf{y} \tag{21}$$

$$\text{s.t. } (\mathbf{e}_1 + \mathbf{e}_2)^T \mathbf{u} = 1, \tag{22}$$

$$B\mathbf{y} - C\mathbf{u} = \mathbf{0}, \tag{23}$$

$$\mathbf{u} \in \mathcal{K}_{r+2}, \tag{24}$$

$$Y \succeq 0, \tag{25}$$

which is an instance of the SQLP problem (Tütüncü et al., 2003). Note that in this formulation, we have traded the semidefinite cone constraint of size $(m^2 + 1) \times (m^2 + 1)$ in (16) with one second-order cone constraint of size $r + 2$ and $r + 1$ linear constraints. As it turns out, such a formulation is much easier to process numerically and can be solved much more efficiently. Indeed, using standard interior-point algorithms, the number of iterations required is of the order $\sqrt{m}$ (Ben-Tal & Nemirovski, 2001, Lecture 6), and the total number of arithmetic operations required is of the order $m^{6.5}$ (Tütüncü et al., 2003). This compares very favorably with the $m^9$ arithmetic complexity of the SCSDP approach, and our experimental results indicate that the speedup in computation is quite substantial. Moreover, in contrast with the SCSDP formulation, which does not take advantage of the low rank structure of $A$, our formulation does take advantage of such a structure.

## 4  Experimental Results

In this section, we perform several experiments to demonstrate the viability of our SQLP formulation and its superior computational performance. Since both the SQLP formulation and the previous SCSDP formulation can be solved by standard softwares to a satisfying gap tolerance, the focus in this comparison is not on the accuracy aspect but on the computational efficiency aspect.

We set the relative gap tolerance for both algorithms to be 1e-08. We use SDPT3 (Toh et al., 2006; Tütüncü et al., 2003) to solve the SQLP. We follow (Weinberger et al., 2007; Li et al., 2009) and use CSDP 6.0.1 (Borchers, 1999) to solve the SCSDP. All experiments are conducted in Matlab 7.6.0(R2008a) on a PC with 2.5GHz CPU and 4GB RAM.

Two benchmark databases, Swiss Roll[3] and USPS[4] are used in our experiments. Swiss Roll (Figure 1(a)) is a standard manifold model used for manifold learning and nonlinear dimensionality reduction. In the experiments, we use the data set shown in Figure 1(b), which consists of 2000 points sampled from the Swiss Roll manifold. USPS is a handwritten digits database widely used for clustering and classification. It contains images of handwritten digits from 0 to 9 of size $16 \times 16$, and has 7291 training examples and 2007 test examples. In the experiments, we use a subset of USPS with 2000 images, containing the first 200 examples of each digit from 0-9 in the training data. The feature to represent each image is a vector formed by concatenating all the columns of the image intensities. In the sequel, we shall refer to the two subsets used in the experiments simply as Swiss Roll and USPS.

Table 1: Computational Results on Swiss Roll (Time: second; # Iter: number of iterations)

| | SCSDP | | | SQLP | | | |
|---|---|---|---|---|---|---|---|
| $m$ | Time | # Iter | Time per Iter | Time | # Iter | Time per Iter | rank($A$) |
| 10 | 3.84 | 29 | 0.13 | 0.96 | 32 | 0.03 | 64 |
| 15 | 60.36 | 30 | 2.01 | 1.75 | 31 | 0.06 | 153 |
| 20 | 557.79 | 32 | 17.43 | 4.48 | 35 | 0.13 | 264 |
| 25 | 2821.76 | 34 | 82.99 | 7.84 | 37 | 0.21 | 403 |
| 30 | 13039.30 | 37 | 352.41 | 13.39 | 35 | 0.38 | 537 |
| 35 | 38559.50 | 33 | 1168.50 | 29.74 | 35 | 0.85 | 670 |
| 40 | > 1 day | — | — | 74.01 | 35 | 2.12 | 852 |
| 50 | — | — | — | 213.26 | 35 | 6.09 | 1152 |
| 60 | — | — | — | 467.90 | 35 | 13.37 | 1451 |
| 80 | — | — | — | 1729.65 | 39 | 44.35 | 2062 |
| 100 | — | — | — | 3988.31 | 36 | 110.79 | 2623 |

Table 2: Computational Results on USPS (Time: second; # Iter: number of iterations)

| | SCSDP | | | SQLP | | | |
|---|---|---|---|---|---|---|---|
| $m$ | Time | # Iter | Time per Iter | Time | # Iter | Time per Iter | rank($A$) |
| 10 | 2.84 | 21 | 0.14 | 0.47 | 16 | 0.03 | 100 |
| 15 | 42.96 | 22 | 1.95 | 1.26 | 17 | 0.07 | 225 |
| 20 | 461.38 | 27 | 17.09 | 3.35 | 17 | 0.20 | 400 |
| 25 | 2572.72 | 31 | 82.99 | 5.97 | 14 | 0.43 | 625 |
| 30 | 10576.01 | 30 | 352.53 | 15.72 | 19 | 0.83 | 900 |
| 35 | 35173.60 | 30 | 1172.50 | 44.53 | 17 | 2.62 | 1225 |
| 40 | > 1 day | — | — | 133.58 | 20 | 6.68 | 1600 |
| 50 | — | — | — | 362.24 | 16 | 22.64 | 2379 |
| 60 | — | — | — | 936.58 | 19 | 49.29 | 2938 |
| 80 | — | — | — | 1784.12 | 17 | 104.95 | 3112 |
| 100 | — | — | — | 2900.44 | 17 | 170.61 | 3111 |

The Swiss Roll (resp. USPS) is used to derive the QSDP in RegMVU (resp. RegPCP). For RegMVU, the 4NN graph is used, i.e., $w_{ij} = 1$ if $\mathbf{x}_i$ is within the 4NN of $\mathbf{x}_j$ or vice versa, and $w_{ij} = 0$ otherwise. We verified that the 4NN graph derived from our Swiss Roll data is connected. For RegPCP, we construct the graph following the approach suggested in (Li et al., 2009). Specifically, we have $w_{ij} = \exp(-d_{ij}^2/(2\sigma^2))$ if $\mathbf{x}_i$ is within 20NN of $\mathbf{x}_j$ or vice versa, and $w_{ij} = 0$ otherwise. Here, $\sigma$ is the averaged distance from each object to its 20-th nearest neighbor. For the pairwise constraints used in RegPCP, we randomly generate 20 similarity constraints for each class, and 20 dissimilarity constraints for every two classes, yielding a total of 1100 constraints. For each data set, $m$ ranges over $\{10, 15, 20, 25, 30, 35, 40, 50, 60, 80, 100\}$. In summary, for each data set, 11 QSDPs are formed. Each QSDP gives rise to one SQLP and one SCSDP. Therefore, for each data set, 11 SQLPs and 11 SCSDPs are derived.

## 4.1 The Results

The computational results of the programs are shown in Tables 1 and 2. For each program, we report the total computation time, the number of iterations needed to achieve the required tolerance, and the average time per iteration in solving the program. A dash (—) in the box indicates that the corresponding program takes too much time to solve. We choose to stop the program if it fails to converge within 1 day. This happens for the SCSDP with $m = 40$ on both data sets.

¿From the tables, we see that solving an SQLP is consistently much more faster than solving an SCSDP. To see the scalability, we plot the solution time (Time) against the problem size ($m$) in Figure 2. It can be seen that the solution time of the SCSDP grows much faster than that of the SQLP. This demonstrates the superiority of our proposed approach.

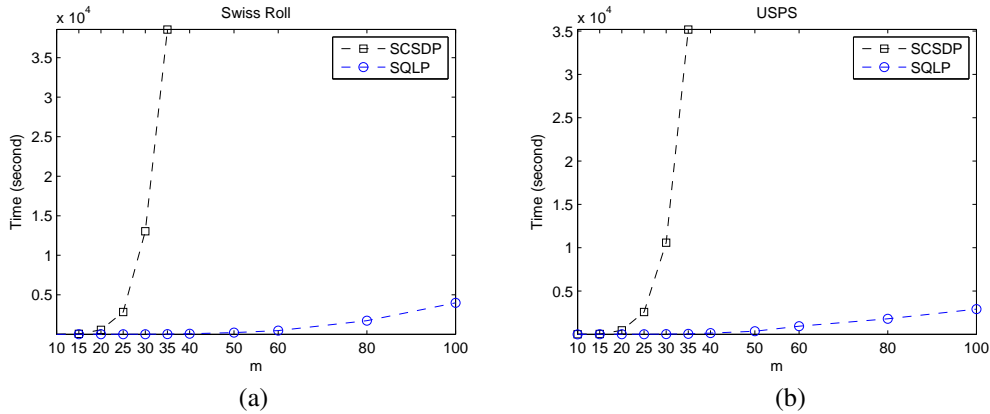

Figure 2: Curves on computational cost: Solution Time vs. Problem Scale.

We also note that the per-iteration computational costs of SCSDP and SQLP are drastically different. Indeed, for the same problem size $m$, it takes much less time per iteration for the SQLP than that for the SCSDP. This is not very surprising, as the SQLP formulation takes advantage of the low rank structure of the data matrix $A$.

## 5  Conclusions

We have studied a class of convex optimization programs called convex *Quadratic Semidefinite Programs* (QSDPs), which arise naturally from graph Laplacian regularized kernel learning (Sha & Saul, 2005; Weinberger et al., 2007; Li et al., 2009; Song et al., 2008; Globerson & Roweis, 2007; Singer, 2008). A QSDP is similar to a QP, except that it is subject to a semidefinite cone constraint as well. To tackle the QSDP, one typically uses the Schur complement to rewrite it as an SDP (SCSDP), thus resulting in a large linear matrix inequality constraint. In this paper, we argue that this formulation is not computationally optimal and have proposed a novel formulation that leads to a semidefinite-quadratic-linear program (SQLP). Our formulation introduces one positive semidefinite constraint, one second-order cone constraint and a set of linear constraints. This should be contrasted with the two large semidefinite cone constraints in the SCSDP. Our complexity analysis and experimental results have shown that the proposed SQLP formulation scales far better than the SCSDP formulation.

## Acknowledgements

The authors would like to thank Professor Kim-Chuan Toh for his valuable comments. This research work was supported in part by GRF grants CUHK 2150603, CUHK 414307 and CRF grant CUHK2/06C from the Research Grants Council of the Hong Kong SAR, China, as well as the NSFC-RGC joint research grant N_CUHK411/07.

## Footnotes

[1] `https://projects.coin-or.org/Csdp/`

[2] `http://www.math.nus.edu.sg/~mattohkc/sdpt3.html`

[3] http://www.cs.toronto.edu/~roweis/lle/code.html

[4] http://www-stat.stanford.edu/~tibs/ElemStatLearn/

## References

Ben-Tal, A., & Nemirovski, A. (2001). *Lectures on Modern Convex Optimization: Analysis, Algorithms, and Engineering Applications*. MPS–SIAM Series on Optimization. Philadelphia, Pennsylvania: Society for Industrial and Applied Mathematics.

Borchers, B. (1999). CSDP, a C Library for Semidefinite Programming. *Optimization Methods and Software*, *11/12*, 613–623.

Boyd, S., & Vandenberghe, L. (2004). *Convex Optimization*. Cambridge: Cambridge University Press. Available online at http://www.stanford.edu/~boyd/cvxbook/.

Chapelle, O., & Vapnik, V. (2000). Model Selection for Support Vector Machines. In S. A. Solla, T. K. Leen and K.-R. Müller (Eds.), *Advances in Neural Information Processing Systems 12: Proceedings of the 1999 Conference*, 230–236. Cambridge, Massachusetts: The MIT Press.

Globerson, A., & Roweis, S. (2007). Visualizing Pairwise Similarity via Semidefinite Programming. *Proceedings of the 11th International Conference on Artificial Intelligence and Statistics* (pp. 139–146).

Golub, G. H., & Loan, C. F. V. (1996). *Matrix Computations*. Baltimore, Maryland: The Johns Hopkins University Press. Third edition.

Graepel, T. (2002). Kernel Matrix Completion by Semidefinite Programming. *Proceedings of the 12th International Conference on Artificial Neural Networks* (pp. 694–699). Springer–Verlag.

Kulis, B., Sustik, M. A., & Dhillon, I. S. (2009). Low–Rank Kernel Learning with Bregman Matrix Divergences. *The Journal of Machine Learning Research*, *10*, 341–376.

Lanckriet, G. R. G., Cristianini, N., Bartlett, P., El Ghaoui, L., & Jordan, M. I. (2004). Learning the Kernel Matrix with Semidefinite Programming. *The Journal of Machine Learning Research*, *5*, 27–72.

Li, Z., & Liu, J. (2009). Constrained Clustering by Spectral Kernel Learning. *To appear in the Proceedings of the 12th IEEE International Conference on Computer Vision*.

Li, Z., Liu, J., & Tang, X. (2008). Pairwise Constraint Propagation by Semidefinite Programming for Semi–Supervised Classification. *Proceedings of the 25th International Conference on Machine Learning* (pp. 576–583).

Li, Z., Liu, J., & Tang, X. (2009). Constrained Clustering via Spectral Regularization. *Proceedings of the IEEE Conference on Computer Vision and Pattern Recognition 2009* (pp. 421–428).

Schölkopf, B., & Smola, A. J. (2002). *Learning with Kernels: Support Vector Machines, Regularization, Optimization, and Beyond*. Cambridge, Massachusetts: The MIT Press.

Schölkopf, B., Smola, A. J., & Müller, K.-R. (1998). Nonlinear Component Analysis as a Kernel Eigenvalue Problem. *Neural Computation*, *10*, 1299–1319.

Sha, F., & Saul, L. K. (2005). Analysis and Extension of Spectral Methods for Nonlinear Dimensionality Reduction. *Proceedings of the 22nd International Conference on Machine Learning* (pp. 784–791).

Shawe-Taylor, J., & Cristianini, N. (2004). *Kernel Methods for Pattern Analysis*. Cambridge: Cambridge University Press.

Singer, A. (2008). A Remark on Global Positioning from Local Distances. *Proceedings of the National Academy of Sciences*, *105*, 9507–9511.

So, A. M.-C. (2007). *A Semidefinite Programming Approach to the Graph Realization Problem: Theory, Applications and Extensions*. Doctoral dissertation, Stanford University.

Song, L., Smola, A., Borgwardt, K., & Gretton, A. (2008). Colored Maximum Variance Unfolding. In J. C. Platt, D. Koller, Y. Singer and S. Roweis (Eds.), *Advances in Neural Information Processing Systems 20: Proceedings of the 2007 Conference*, 1385–1392. Cambridge, Massachusetts: The MIT Press.

Toh, K. C., Tütüncü, R. H., & Todd, M. J. (2006). On the Implementation and Usage of SDPT3 — A MATLAB Software Package for Semidefinite–Quadratic–Linear Programming, Version 4.0. User's Guide.

Tütüncü, R. H., Toh, K. C., & Todd, M. J. (2003). Solving Semidefinite–Quadratic–Linear Programs using SDPT3. *Mathematical Programming*, *95*, 189–217.

Vapnik, V. N. (2000). *The Nature of Statistical Learning Theory*. Statistics for Engineering and Information Science. New York: Springer–Verlag. Second edition.

Weinberger, K. Q., Packer, B. D., & Saul, L. K. (2005). Nonlinear Dimensionality Reduction by Semidefinite Programming and Kernel Matrix Factorization. *Proceedings of the 10th International Workshop on Artificial Intelligence and Statistics* (pp. 381–388).

Weinberger, K. Q., Sha, F., & Saul, L. K. (2004). Learning a Kernel Matrix for Nonlinear Dimensionality Reduction. *Proceedings of the 21st International Conference on Machine Learning* (pp. 85–92).

Weinberger, K. Q., Sha, F., Zhu, Q., & Saul, L. K. (2007). Graph Laplacian Regularization for Large–Scale Semidefinite Programming. *Advances in Neural Information Processing Systems 19: Proceedings of the 2006 Conference* (pp. 1489–1496). Cambridge, Massachusetts: The MIT Press.

Zhou, D., Bousquet, O., Lal, T. N., Weston, J., & Schölkopf, B. (2004). Learning with Local and Global Consistency. *Advances in Neural Information Processing Systems 16: Proceedings of the 2003 Conference* (pp. 595–602). Cambridge, Massachusetts: The MIT Press.

